# A Neuromorphic Monaural Sound Localizer

**John G. Harris, Chiang-Jung Pu, and Jose C. Principe**
Department of Electrical & Computer Engineering
University of Florida
Gainesville, FL 32611

## Abstract

We describe the first single microphone sound localization system and its inspiration from theories of human monaural sound localization. Reflections and diffractions caused by the external ear (pinna) allow humans to estimate sound source elevations using only one ear. Our single microphone localization model relies on a specially shaped reflecting structure that serves the role of the pinna. Specially designed analog VLSI circuitry uses echo-time processing to localize the sound. A CMOS integrated circuit has been designed, fabricated, and successfully demonstrated on actual sounds.

## 1 Introduction

The principal cues for human sound localization arise from time and intensity differences between the signals received at the two ears. For *low-frequency* components of sounds (below 1500Hz for humans), the phase-derived interaural time difference (ITD) can be used to localize the sound source. For these frequencies, the sound wavelength is at least several times larger than the head and the amount of shadowing (which depends on the wavelength of the sound compared with the dimensions of the head) is negligible. ITD localization is a well-studied system in biology (see e.g., [5]) and has even been mapped to neuromorphic analog VLSI circuits with limited success on actual sound signals [6] [2]. Above 3000Hz, interaural phase differences become ambiguous by multiples of $360°$ and are no longer viable localization cues. For these high frequencies, the wavelength of the sound is small enough that the sound amplitude is attenuated by the head. The intensity difference of the log magnitudes at the ears provides a unique interaural intensity difference (IID) that can be used to localize.

Many studies have shown that when one ear is completely blocked, humans can still localize sounds in space, albeit at a worse resolution in the horizontal direc-

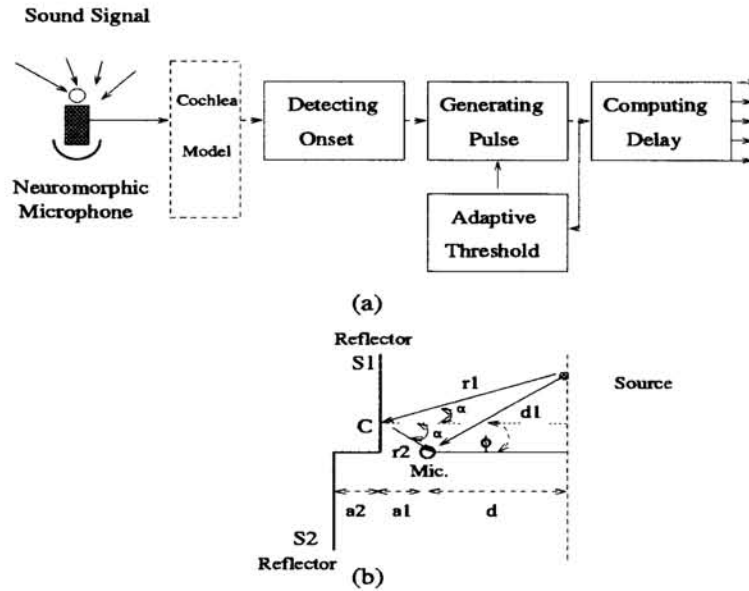

Figure 1: *(a) Proposed localization model is inspired from the biological model (b) Special reflection surface to serve the role of the pinna*

tion. Monaural localization requires that information is somehow extracted from the direction-dependent effects of the reflections and diffractions of sound off of the external ear (pinna), head, shoulder, and torso. The so-called "Head Related Transfer Function" (HRTF) is the effective direction-dependent transfer function that is applied to the incoming sound to produce the sound in the middle ear. Section 2 of this paper introduces our monaural sound localization model and Section 3 discusses the simulation and measurement results.

## 2   Monaural Sound Localization Model

Batteau [1] was one of the first to emphasize that the external ear, specifically the pinna, could be a source of spatial cues that account for vertical localization. He concluded that the physical structure of the external ear introduced two significant echoes in addition to the original sound. One echo varies with the azimuthal position of the sound source, having a latency in the 0 to $80\mu s$ range, while the other varies with elevation in the $100\mu s$ to $300\mu s$ range. The output $y(t)$ at the inner ear is related to the original sound source $x(t)$ as

$$y(t) = x(t) + a_1 x(t - \tau_a) + a_2 x(t - \tau_v)$$   (1)

where $\tau_a, \tau_v$ refer to azimuth and elevation echoes respectively; $a_1$ and $a_2$ are two reflection constants. Other researchers subsequently verified these results [11] [4].

Our localizer system (shown in Figure 1(a)) is composed of a special reflection surface that encodes the sound source's direction, a silicon cochlea that functions as a band-pass filter bank, onset detecting circuitry that detects and amplifies the energy change at each frequency tap, pulse generating circuitry that transfers analog sound signals into pulse signals based on adaptively thresholding the onset signal, and delay time computation circuitry that computes the echo's time delay then decodes the sound source's direction.

Since our recorded signal is composed of a direct sound and an echo, the sound is a simplified version of actual HRTF recordings that are composed of the direct sound

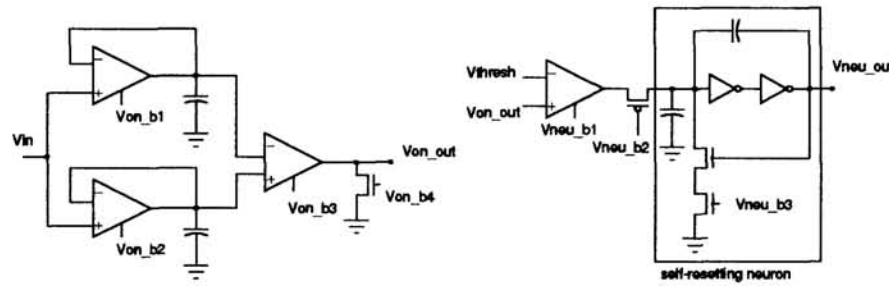

Figure 2: *(a) Sound signal's onset is detected by taking the difference of two low-pass filters with different time constants. (b) Pulse generating circuit.*

and its reflections from the external ear, head, shoulder, and torso. To achieve localization in a 1D plane, we may use any shape of reflection surface as long as the reflection echo caused by the surface provides a one-to-one mapping between the echo's delay time and the source's direction. Thus, we propose two flat surfaces to compose the reflection structure in our proposed model depicted in Figure 1(b). A microphone is placed at distances $a_1$ and $a_2$ from two flat surfaces ($S_1$ and $S_2$), $d$ is the distance between the microphone and the sound source moving line (the dotted line in Figure 1(b). As shown in Figure 1(b), a sound source is at $\angle\phi$ position. If the source is far enough from the reflection surface, the ray diagram is valid to analyze the sound's behavior. We skip the complete derivation but the echo's delay time can be expressed as

$$\tau = \frac{r_1 + r_2 - d_1}{c} \tag{2}$$

where $d_1$ is the length of the direct path, $r_1 + r_2$ is reflected path length, and $c$ is the speed of sound. The path distance are easily solved in terms of the source direction and the geometry of the setup (see [9] for complete details).

The echo's delay time $\tau$ decreases as the source position $\phi$ moves from 0 to 90 degrees. A similar analysis can be made if the source moves in the opposite direction, and the reflection is caused by the other reflection surface $S_2$. Since the reflection path is longer for reflection surface $S_2$ than for reflection surface $S_1$, the echo's delay time can be segmented into two ranges. Therefore, the echo's delay time encodes the source's directions in a one-to-one mapping relation.

In the setup, an Earthworks M30 microphone and Lab1 amplifier were used to record and amplify the sound signals [3]. For this preliminary study of monaural localization, we have chosen to localize simple impulse sounds generated through speakers and therefore can drop the silicon cochlea from our model. In the future, more complicated signals, such as speech, will require a silicon cochlea implementation.

Inspired by ideas from visual processing, onset detection is used to segment sounds [10]. The detection of an onset is produced by first taking the difference of two first-order, low-pass filters given by [10]

$$O(t, k, r) = \int_0^t f_z(t - x, k)s(x)dx - \int_0^t f_z(t - x, k/r)s(x)dx \tag{3}$$

where r>1, $k$ is a time constant, $s(x)$ is the input sound signal, and $f_z(x, k) = k \exp(-kx)$.

A hardware implementation of the above equation is depicted in Figure 2a. In our model, sound signals from the special reflection surface microphone are fed into two low-pass filters which have different time constants determined by two bias

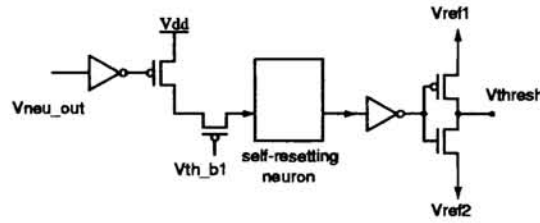

Figure 3: *Adaptive threshold circuit used to remove unwanted reflections.*

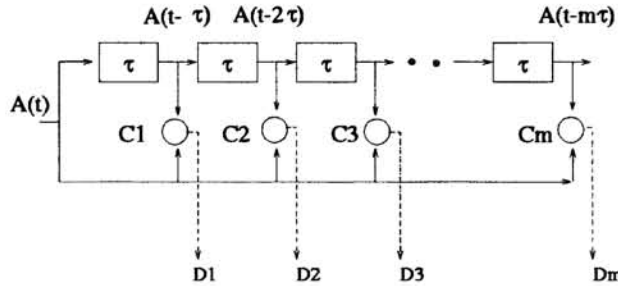

Figure 4: *Neural signal processing model*

voltages $V_{on_{b1}}$ and $V_{on_{b2}}$. The bias voltage $V_{on_{b3}}$ determines the amplification of the difference. The output of the onset detecting circuit is $V_{on_{out}}$. The onset detection circuit determines significant increases in the signal energy and therefore segments sound events. By computing the delay time between two sound events (direct sound and its echo caused by the reflection surface), the system is able to decode the source's direction. Each sound event is then transformed into a fixed-width pulse so that the delay time can be computed with binary autocorrelators.

The fixed-width pulse generating circuit is depicted in Figure 2b. The pulse generating circuit includes a self-resetting neuron circuit [8] that controls the pulse duration based on the bias voltage $V_{neu_{b3}}$. As discussed above, an appropriate threshold is required to discriminate sound events from noise. One input of the pulse generating circuit is the output of the onset detecting signal, $V_{on_{out}}$. $V_{thresh}$ is set properly in the pulse generating circuit in order to generate a fixed width pulse when $V_{on_{out}}$ exceeds $V_{thresh}$. Unfortunately the system may be confused by unwanted sound events due to extraneous reflections from the desks and walls. However, since we know the expect range of echo delays, we can inhibit many of the environmental echoes that fall outside this range using an adaptive threshold circuit.

In order to cancel unwanted signals, we need to design an inhibition mechanism which suppresses signals arriving to our system outside of the expected time range. This inhibition is implemented in Figure 3. As the pulse generating circuit detects the first sound event (which is the direct sound signal), the threshold becomes high in a certain period of time to suppress the detection of the unwanted reflections (not from our reflection surfaces). The input of the adaptive threshold circuit is $V_{neu_{out}}$ which is the output of the pulse generating circuit. The output of the threshold circuit is $V_{thresh}$ which is the input of the pulse generating circuit. When the pulse generating circuit detects a sound event, $V_{neu_{out}}$ becomes high, which increases $V_{thresh}$ from $V_{ref2}$ to $V_{ref1}$ as shown in Figure 3. The higher $V_{thresh}$ suppresses the detection. The suppression time is determined by the other self-resetting neuron circuit.

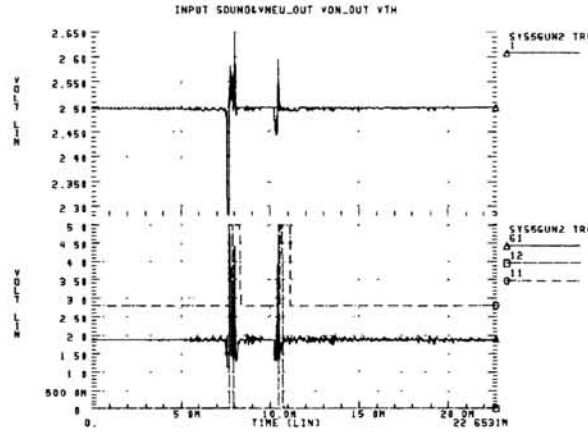

Figure 5: *(a) The input sound signal: impulse signal recorded in typical office environment (b) HSPICE simulation of the output of the detecting onset circuit (label 61), the output of the pulse generating circuit (label 12), and the adaptive threshold circuit response (label 11)*

The nervous system likely uses a running autocorrelation analysis to measure the time delay between signals. The basic neural connections are shown in Figure 4 [7]. $A(t)$ is the input neuron, $A(t-\tau)$, $A(t-2\tau)$,...$A(t-m\tau)$ is a delay chain. The original signal and the delayed signal are multiplied when $A(t)$ and $A(t-k\tau)$ feed $C_k$. Assuming the state of neuron A is $N_A(t)$. If each synaptic delay in the chain is $\tau$, the chain gives us $N_A(t)$ under various delays. $C_k$ fires simultaneously when both $A(t)$ and $A(t-k\tau)$ fire. Neuron $C_k$ connects neuron $D_k$. Excitation is built up at $D_k$ by the charge and discharge of $C_k$. The excitation at $D_k$ is therefore

$$D_k(t) = N_{C_k}(t) = N_A(t)N_A(t-k\tau) \qquad (4)$$

Viewing the arrangement of Figure 4 as a neuron autocorrelator, the time-varying excitation at $D_1, D_2, ..D_k$ provides a spatial representation of the autocorrelation function. The localization resolution of this system depends on the delay time $\tau$, and the number of the correlators. As $\tau$ decreases, the localization resolution is improved provided there are enough correlators. In this paper, 30 unit delay taps, and 10 correlators have been implemented on chip. The outputs of the 10 correlators display the time difference between two sound events. The delay time decodes the source's direction. Therefore, the 10 correlators provide a unit encoding of the source location in the 1D plane.

## 3   Simulation and Measurement Results

The complete system has been successfully simulated in HSPICE using database we have recorded. Figure 5(a) shows the input sound signal which is an impulse signal recording in our lab (a typical student office environment). Figure 5(b) shows the output of the onset detector (labeled 61), the pulse generating output (labeled 12), and the adaptive threshold (labeled 11). When the onset output exceeds the threshold, the output of the pulse generating circuit becomes high. Simultaneously, the high value of the generated pulse turns on the adaptive threshold circuit to increase the threshold voltage. The adaptive threshold voltage suppresses the unwanted re-

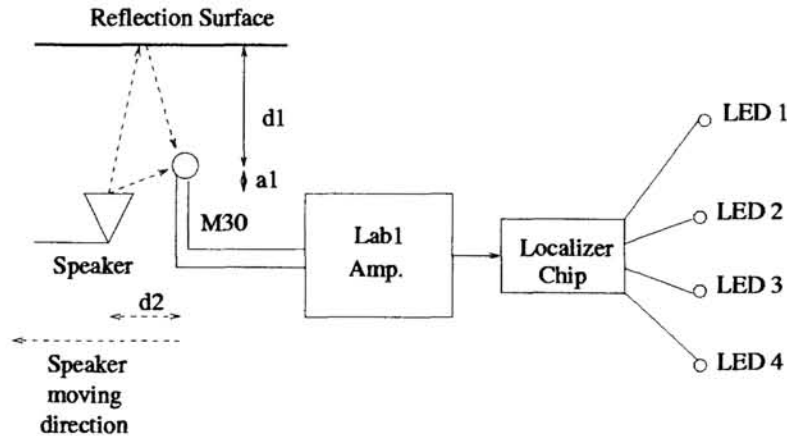

Figure 6: *Block diagram of the test setup*

flection which can be seen right after the direct signal (we believe the unwanted reflection is caused by the table). Further simulation results are discussed in [9].

The single microphone sound localizer circuit has been fabricated through the MO-SIS $2\mu$m N-well CMOS process. Impulse signals are played through speakers to test the fabricated localizer chip. Figure 6 depicts the block diagram of the test setup. The M30 microphone picks up the direct impulse signal and echoes from the reflection surface. Since the reflection surface in our test is just a single flat surface, localization is only tested in one-half of the 1D plane. The composite signals are fed into the input of the sound localizer after amplification. Our sound localizer chip receives the composite signal, computes the echo time delay, and sends out the localization result to a display circuit. The display circuit is composed of 4 LEDs with each LED representing a specific sound source location. The sound localizer sends the computational result to turn on a specific LED signifying the echo time delay. In the test, the M30 microphone and the reflection surface are placed at fixed locations. The speaker is moved along the dotted line shown in Figure 6. The M30 microphone is $d_1$ (33cm) from the reflection surface and $a_1$ (24cm) from the speaker moving line. The speaker's location is defined as $d_2$ as depicted in Figure 6.

Figure 7(a) shows the theoretical echo's delay at various speaker locations. Figure 7(b) is the measurement of the setup depicted in Figure 6. The y-axis indicates LED 1 through LED 4. The x-axis represents the distance between the speaker's location ($d_2$ in Figure 6). The solid horizontal line in Figure 7(b) represents the theoretical results for which LED should respond for each displacement. The results show that localization is accurate within each region with possibilities of two LEDs responding in the overlap regions.

## 4  Conclusion

We have developed the first monaural sound localization system. This system provides a real-time model for human sound localization and has potential use in such applications as low-cost teleconferencing. More work is needed to further develop the system. We need to characterize the accuracy of our system and to test more interesting sound signals, such as speech. Our flat reflection surface is straightforward and simple, but it lacks sufficient flexibility to encode the source's direction in more than a 1-D plane. We plan to replace the flat surfaces with a more complicated surface to provide more reflections to encode a richer set of source directions.

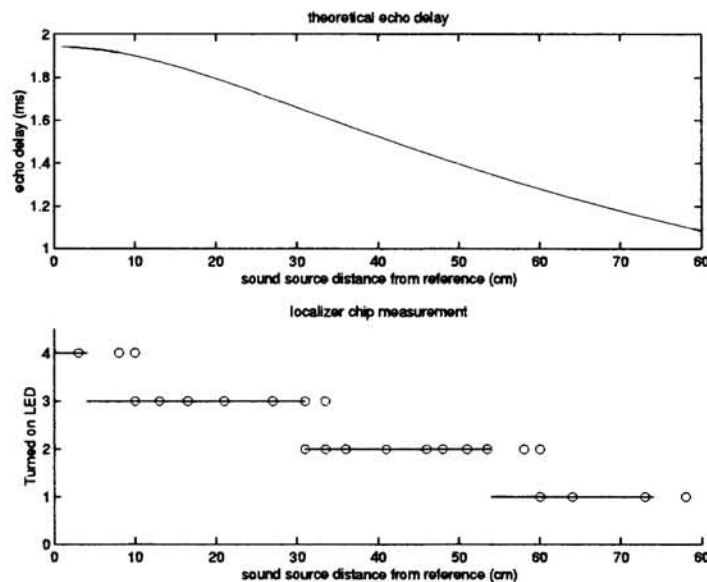

Figure 7: *Sound localizer chip test result*

## Acknowledgments

This work was supported by an ONR contract #N00014-94-1-0858 and an NSF CA-REER award #MIP-9502307. We gratefully acknowledge MOSIS chip fabrication and Earthworks Inc. for loaning the M30 microphone and amplifier.

## References

[1] D. W. Batteau. The role of the pinna in human localization. *Proc. R. Soc. London, Ser. B*, 168:158–180, 1967.

[2] Neal A. Bhadkamkar. Binaural source localizer chip using subthreshold analog cmos. In *Proceeding of ICNN*, pages 1866–1870, 1994.

[3] Earthworks, Inc., P.O. Box 517, Wilton, NH 03086. *M30 Microphone.*

[4] Y. Hiranaka and H. Yamasaki. Envelop representations of pinna impulse responses relating to three-dimensional localization of sound sources. *J. Acoust. Soc. Am.*, 73:29, 1983.

[5] E. Knudsen, G. Blasdel, and M. Konishi. Mechanisms of sound localization in the barn owl (tyto alba). *J. Comp. Physiol*, 133:13–21, 1979.

[6] J. Lazzaro and C. A. Mead. A silicon model of auditory localization. *Neural Computation*, 1:47–57, 1989.

[7] J.C. Licklider. A duplex theory of pitch perception. *Experientia*, 7:128–133, 1951.

[8] C. Mead. *Analog VLSI and Neural Systems*. Addison-Wesley, 1989.

[9] Chiang-Jung Pu. *A neuromorphic microphone for sound localization.* PhD thesis, University of Florida, Gainesville, FL, May 1998.

[10] L.S. Smith. Sound segmentation using onsets and offsets. *J. of New Music Research*, 23, 1994.

[11] A.J. Watkins. Psychoacoustical aspects of synthesized vertical locale cues. *J. Acoust. Soc. Am.*, 63:1152–1165, 1978.